# (Not) Bounding the True Error

**John Langford**
Department of Computer Science
Carnegie-Mellon University
Pittsburgh, PA 15213
*jcl+@cs.cmu.edu*

**Rich Caruana**
Department of Computer Science
Cornell University
Ithaca, NY 14853
*caruana@cs.cornell.edu*

## Abstract

We present a new approach to bounding the true error rate of a continuous valued classifier based upon PAC-Bayes bounds. The method first constructs a distribution over classifiers by determining how sensitive each parameter in the model is to noise. The true error rate of the stochastic classifier found with the sensitivity analysis can then be tightly bounded using a PAC-Bayes bound. In this paper we demonstrate the method on artificial neural networks with results of a $2 - 3$ order of magnitude improvement vs. the best deterministic neural net bounds.

## 1 Introduction

In machine learning it is important to know the true error rate a classifier will achieve on future test cases. Estimating this error rate can be suprisingly difficult. For example, all known bounds on the true error rate of artificial neural networks tend to be extremely loose and often result in the meaningless bound of "always err" (error rate = 1.0).

In this paper, we do *not* bound the true error rate of a neural network. Instead, we bound the true error rate of a distribution over neural networks which we create by analysing one neural network. (Hence, the title.) This approach proves to be much more fruitful than trying to bound the true error rate of an individual network. The best current approaches [1][2] often require 1000, 10000, or more examples before producing a nontrivial bound on the true error rate. We produce nontrivial bounds on the true error rate of a stochastic neural network with less than 100 examples. A stochastic neural network is a neural network where each weight $w_i$ is perturbed by a gaussian with variance $s_i^2$ *every* time it is evaluated.

Our approach uses the PAC-Bayes bound [5]. The approach can be thought of as a redivision of the work between the experimenter and the theoretician: we make the experimenter work harder so that the theoretician's true error bound becomes much tighter. This "extra work" on the part of the experimenter is significant, but tractable, and the resulting bounds are *much* tighter.

An alternative viewpoint is that the classification problem *is* finding a hypothesis with a low upper bound on the future error rate. We present a post-processing phase for neural networks which results in a classifier with a much lower upper bound on the future error rate. The post-processing can be used with any artificial neural net trained with any optimization method; it does not require the learning procedure be modified, re-run, or even that the threshold function be differentiable. In fact, this post-processing step can easily be adapted to other learning algorithms.

David MacKay [4] has done significant work to make approximate Bayesian learning tractable with a neural network. Our work here is complimentary rather than competitive. We exhibit a technique which will likely give nontrivial true error rate bounds for Bayesian

neural networks *regardless* of approximation or prior modeling errors. Verification of this statement is work in progress.

The post-processing step finds a "large" distribution over classifiers, which has a small *average* empirical error rate. Given the average empirical error rate, it is straightforward to apply the PAC-Bayes bound in order to find a bound on the *average* true error rate. We find this large distribution over classifiers by performing a simple noise sensitivy analysis on the learned model. The noise model allows us to generate a distribution of classifiers with a known, small, average empirical error rate. In this paper we refer to the distribution of neural nets that results from this noise analysis as a stochastic neural net model.

Why do we expect the PAC-Bayes bound to be a significant improvement over standard covering number and VC bound approaches? There exist learning problems for which the difference between the lower bound and the PAC-Bayes upper bound are tight up to $O\left(\frac{\ln m}{m}\right)$ where $m$ is the number of training examples. This is superior to the guarantees which can be made for typical covering number bounds where the gap is, at best, known up to an (asymptotic) constant. The guarantee that PAC-Bayes bounds are sometimes quite tight encourages us to apply them here.

The next sections will:

1. Describe the bounds we will compare.

2. Describe our algorithm for constructing a distribution over neural networks.

3. Present experimental results.

## 2   Theoretical setup

We will work in the standard supervised batch learning setting. This setting starts with the assumption that all examples are drawn from some fixed (unknown) distribution, $D$, over *(input, output)* pairs, $(x, y)$. The output $y$ is drawn from the space $\{-1, 1\}$ and the input space is arbitrary. The goal of machine learning is to use a sample set $S$ of $m$ pairs to find a classifier, $h$, which maps the input space to the output space and has a small true error, $e(h) \equiv \Pr_D(h(x) \neq y)$. Since the distribution $D$ is unknown, the true error rate is not observable. However, we can observe the empirical error rate, $\hat{e}(h) \equiv \Pr_S(h(x) \neq y) = \frac{1}{m} \sum_{i=1}^m h(x_i) \neq y_i$.

Now that the basic quantities of interest are defined, we will first present a modern neural network bound, then specialize the PAC-Bayes bound to a stochastic neural network. A stochastic neural network is simply a neural network where each weight in the neural network is drawn from some distribution whenever it is used. We will describe our technique for constructing the distribution of the stochastic neural network.

### 2.1   Neural Network bound

We will compare a specialization of the best current neural network true error rate bound [2] with our approach. The neural network bound is described in terms of the following parameters:

1. A margin, $0 < \gamma < 1$.

2. An *arbitrary* function (unrelated to the neural network sigmoid function) $\phi$ defined by $\phi(x) = 1$ if $x < 0$, $\phi(x) = 0$ if $x > 1$, and linear in between.

3. $A_i$, an upper bound on the sum of the magnitude of the weights in the $i$th layer of the neural network

4. $L_i$, a Lipschitz constant which holds for the $i$th layer of the neural network. A Lipschitz constant is a bound on the magnitude of the derivative.

5. $d$, the size of the input space.

With these parameters defined, we get the following bound.

**Theorem 2.1** *(2 layer feed-forward Neural Network true error bound)*

$$\Pr_D\left(\exists h \in H : \ e(h) > \inf_\gamma \ b(\gamma)\right) \leq \delta$$

*where* $b(\gamma) = \frac{1}{m}\sum_{(x,y)}\phi\left(\frac{yh(x)}{\gamma}\right) + \frac{2\sqrt{2\pi}}{\gamma}32\sqrt{\frac{d+1}{m}}L_1L_2A_1A_2 + \frac{\sqrt{\frac{1}{2}\ln\frac{2}{\delta}}+2}{\sqrt{m}}$

**Proof:** Given in [2]. $\square$

The theorem is actually only given up to a universal constant. "32" might be the right choice, but this is just an educated guess. The neural network true error bound above is (perhaps) the tightest known bound for general feed-forward neural networks and so it is the natural bound to compare with.

This 2 layer feed-forward bound is not easily applied in a tight manner because we can't calculate a priori what our weight bound $A_i$ should be. This can be patched up using the principle of structural risk minimization. In particular, we can state the bound for $A_1 = \alpha^j$ where $j$ is some non-negative integer and $\alpha > 1$ is a constant. If the $j$th bound holds with probability $\frac{6}{\pi^2}\frac{\delta}{j^2}$, then all bounds will hold simultaneously with probability $1 - \delta$, since

$$\sum_{j=1}^{\infty}\frac{1}{j^2} = \frac{\pi^2}{6}$$

Applying this approach to the values of both $A_1$ and $A_2$, we get the following theorem:

**Theorem 2.2** *(2 layer feed-forward Neural Network true error bound)*

$$\Pr_{D}\left(\exists h \in H : \ e(h) > \inf_{\gamma j k}\ b(\gamma, j, k)\right) \leq \delta$$

*where* $b(\gamma, j, k) = \frac{1}{m}\sum\phi\left(\frac{yh(x)}{\gamma}\right) + \frac{2\sqrt{2\pi}}{\gamma}32\sqrt{\frac{d+1}{m}}L_1L_2\alpha^j\beta^k + \frac{\sqrt{\frac{1}{2}\ln\frac{\pi^4 j^2 k^2}{36\delta}}+2}{\sqrt{m}}$

**Proof:** Apply the union bound to all possible values of $j$ and $k$ as discussed above. $\square$
In practice, we will use $\alpha = \beta = 1.1$ and report the value of the tightest applicable bound for all $j, k$.

## 2.2 Stochastic Neural Network bound

Our approach will start with a simple refinement [3] of the original PAC-Bayes bound [5]. We will first specialize this bound to stochastic neural networks and then show that the use of this bound in conjunction with a post-processing algorithm results in a much tighter true error rate upper bound.

First, we will need to define some parameters of the theorem.

1. $Q$ is a distribution over the hypotheses which can be found in an example dependent manner.

2. $P$ is a distribution over the hypotheses which is chosen a priori—without dependence on the examples.

3. $e_Q(h) = E_{h\sim Q}e(h)$ is the true error rate of the stochastic hypothesis which, in any evaluation, draws a hypothesis $h$ from $Q$, and outputs $h(x)$.

4. $\hat{e}_Q(h) = E_{h\sim Q}\hat{e}(h)$ is the average empirical error rate of the same stochastic hypothesis.

Now, we are ready to state the theorem.

**Theorem 2.3** *(PAC-Bayes Relative Entropy Bound) For all priors, P,*

$$\Pr_{D}\left(\exists Q : \ KL(\hat{e}_Q(h)\|e_Q(h)) \geq \frac{KL(Q\|P) + \ln\frac{2m}{\delta}}{m - 1}\right) \leq \delta$$

*where $KL(Q\|P) = \int_h q(h)\ln\frac{q(h)}{p(h)}dh$ is the Kullback-Leibler divergence between the distributions $Q$ and $P$ and $KL(\hat{e}_Q(h)\|e_Q(h))$ is the KL divergence between a coin of bias $\hat{e}_Q(h)$ and a coin of bias $e_Q(h)$.*

**Proof:** Given in [3]. □

We need to specialize this theorem for application to a stochastic neural network with a choice of the "prior". Our "prior" will be zero on all neural net structures other than the one we train and a multidimensional isotropic gaussian on the values of the weights in our neural network. The multidimensional gaussian will have a mean of $0$ and a variance in each dimension of $b^2$. This choice is made for convenience and happens to work.

The optimal value of $b$ is unknown and dependent on the learning problem so we will wish to parameterize it in an example dependent manner. We can do this using the same trick as for the original neural net bound. Use a sequence of bounds where $b = c\alpha^j$ for $c$ and $\alpha$ some constants and $j$ a nonnegative number. For the $j$th bound set $\delta \to \frac{6\delta}{\pi^2 j^2}$. Now, the union bound will imply that all bounds hold simultaneously with probability at least $1 - \delta$.

Now, assuming that our "posterior" $Q$ is also defined by a multidimensional gaussian with the mean and variance in each dimension defined by $w_i$ and $s_i^2$, we can specialize to the following corollary:

**Corollary 2.4** *(Stochastic Neural Network bound) Let $k$ be the number of weights in a neural net, $w_i$ be the $i$th weight and $s_i$ be the variance of the $i$th weight. Then, we have*

$$\Pr_D \left( \exists Q : \ KL(\hat{e}_Q(h)||e_Q(h)) \geq \inf_j \frac{\sum_{i=1}^{k}[\ln \frac{c\alpha^j}{s_i} + \frac{s_i^2+w_i^2}{2c^2\alpha^{2j}} - \frac{1}{2}] + \ln \frac{\pi^2 j^2 m}{3\delta}}{m-1} \right) \leq \delta \quad (1)$$

**Proof:** Analytic calculation of the KL divergence between two multidimensional Gaussians and the union bound applied for each value of $j$. □

We will choose $\alpha = 1.1$ and $c = 0.2$ as reasonable default values.

One more step is necessary in order to apply this bound. The essential difficulty is evaluting $\hat{e}_Q(h)$. This quantity is observable although calculating it to high precision is difficult. We will avoid the need for a direct evaluation by a monte carlo evaluation and a bound on the tail of the monte carlo evaluation. Let $\hat{e}_{\hat{Q}}(h) \equiv \Pr_{\hat{Q},S}(h(x) \neq y)$ be the observed rate of failure of a $n$ random hypotheses drawn according to $Q$ and applied to a random training example. Then, the following simple bound holds:

**Theorem 2.5** *(Sample Convergence Bound) For all distributions, $Q$, for all sample sets $S$,*

$$\Pr_Q \left( KL(\hat{e}_{\hat{Q}}(h)||\hat{e}_Q(h)) \geq \frac{\ln \frac{2}{\delta}}{n} \right) \leq \delta$$

*where $n$ is the number of evaluations of the stochastic hypothesis.*

**Proof:** This is simply an application of the Chernoff bound for the tail of a Binomial where a "head" occurs when an error is observed and the bias is $\hat{e}_Q(h)$. □

In order to calculate a bound on the expected true error rate, we will first bound the expected empirical error rate $\hat{e}_Q(h)$ with confidence $\frac{\delta}{2}$ then bound the expected true error rate $e_Q(h)$ with confidence $\frac{\delta}{2}$, using our bound on $\hat{e}_Q(h)$. Since the total probability of failure is only $\frac{\delta}{2} + \frac{\delta}{2} = \delta$ our bound will hold with probability $1 - \delta$. In practice, we will use $n = 1000$ evaluations of the empirical error rate of the stochastic neural network.

## 2.3 Distribution Construction algorithm

One critical step is missing in the description: How do we calculate the multidimensional gaussian, $Q$? The variance of the posterior gaussian needs to be dependent on each weight in order to achieve a tight bound since we want any "meaningless" weights to not contribute significantly to the overall sample complexity. We use a simple greedy algorithm to find the appropriate variance in each dimension.

1. Train a neural net on the examples

2. For every weight, $w_i$, search for the variance, $s_i^2$, which reduces the empirical accuracy of the stochastic neural network by some fixed target percentage (we use $1 - 5\%$) while holding all other weights fixed.

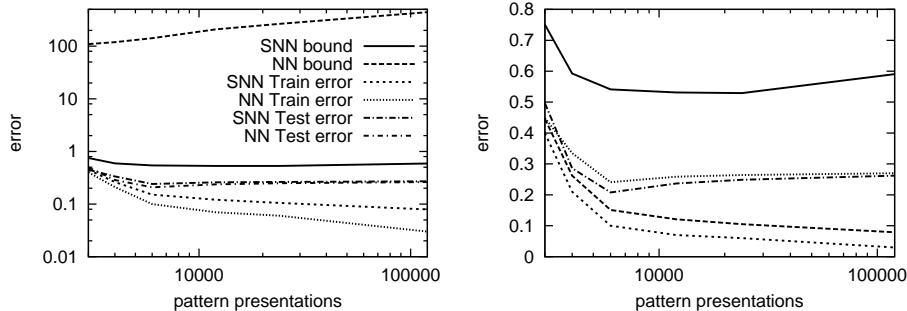

Figure 1: Plot of measured errors and error bounds for the neural network (NN) and the stochastic neural network (SNN) on the synthetic problem. The training set has 100 cases and the reduction in empirical error is 5%. Note that a true error bound of "100" (visible in the graph on the left) implies that at least $100^2$ more examples are required in order to make a nonvacuous bound. The graph on the right expands the vertical scale by excluding the poor true error bound that has error above 100. The curves for NN and SNN are qualitatively similar on the train and test sets. As expected, the SNN consistently performs 5% worse than the NN on the train set (easier to see in the graph on the right). Surprisingly, the SNN performs worse than the NN by less than 5% on the test sets. Both NN and SNN exhibit overfitting after about 6000-12000 pattern presentations (600-1200 epochs). The shape of the SNN bound roughly mimics the shape of the empirically measured true error (this is more visible in the graph on the right) and thus might be useful for indicating where the net begins overfitting.

3. The stochastic neural network defined by $\{w_i,\ s_i^2\}$ will generally have a too-large empirical error. Therefore, we calculate a global multiplier $\lambda < 1$ such that the stochastic neural network defined by $\{w_i,\ \lambda s_i^2\}$ decreases the empirical accuracy by only the same $1 - 5\%$ (absolute error rate) used in Step 2.

4. Then, we evaluate the empirical error rate of the resulting stochastic neural net by repeatedly drawing samples from the stochastic neural network. In the work reported here we use $100 - 1000$ samples.

## 3  Experimental Results

How well can we bound the true error rate of a stochastic neural network? The answer is *much* better than we can bound the true error rate of a neural network.

We use two datasets to empirically evaluate the quality of the new bound. The first is a synthetic dataset which has 25 input dimensions and one output dimension. Most of these dimensions are useless—simply random numbers drawn from a $N(0, 1)$ Gaussian. One of the 25 input dimensions is dependent on the label. First, the label $y$ is drawn uniformly from $\{-1, 1\}$, then the special dimension is drawn from a $N(y, 1)$ Gaussian. Note that this learning problem can not be solved perfectly because some examples will be drawn from the tails where the gaussians overlap. The "ideal" neural net to use in solving this synthetic problem is a single node perceptron. We will instead use a 2 layer neural net with 2 hidden nodes using the sigmoid transfer function. This overly complex neural net will result in the potential for significant overfitting which makes the bound prediction problem interesting. It is also somewhat more "realistic" if the neural net structure does not exactly match the learning problem.

The second dataset is the ADULT problem from the UCI Machine Learning Repository. We use a 2 layer neural net with 2 hidden units for this problem as well because preliminary experiments showed that nets this small can overfit the ADULT dataset if the training sample is small.

To keep things challenging, we use just $100 - 200$ examples in our experiments. As

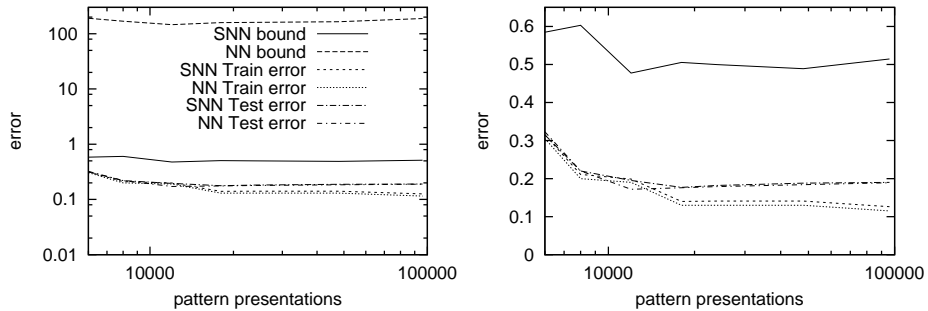

Figure 2: Plot of measured errors and error bounds for the neural network (NN) and the stochastic neural network (SNN) on the UCI ADULT dataset. These graphs show the results obtained using a 1% reduction in empirical error instead of the 5% reduction used in Figure 1. The training sample size for this problem is 200 cases. NN and SNN exhibit overfitting after approximately 12000 pattern presentations (600 epochs). As in Figure 1, a true error bound of "100" implies that at least $100^2$ more examples are required in order to make a nonvacuous bound. The graph on the right expands the vertical scale by excluding the poor true error bound.

we will see in Figures 1 and 2, constructing a nonvacuous bound for a continuous hypothesis space with only $100 - 200$ examples is quite difficult. The conventional bounds are hopelessly loose.

Figure 1 shows the results for the synthetic problem. For this problem we use 100 training cases and a 5% reduction in empirical error. The results for the ADULT problem are presented in Figure 2. For this problem we use 200 training cases and a 1% reduction in empirical error. Experiments performed on these problems using somewhat smaller and larger training samples yielded similar results. The choice of reduction in empirical error is somewhat arbitrary. We see qualitatively similar results if we switch to a 1% reduction for the synthetic problem and a 5% reduction for the ADULT problem.

There are several things worth noting about the results in the two figures.

1. The SNN upper bounds are *2-3* orders of magnitude lower than the NN upper bounds. While not as tight as might be desired, the SNN upper bounds are orders of magnitude better and are not vacuous.

2. The SNNs perform somewhat better than expected. In particular, on the synthetic problem the SNN true error rate is at most $3\%$ worse than the true error rate of the NN (true error rates are estimated using large test sets). This is suprising considering that we fixed the difference in empirical error rates at $5\%$ for the synthetic problem. Similarly, on the ADULT problem we observe that the true error rates between the SNN and NN typically is only about 0.5%, about half of the target difference of 1%. This is good because it suggests that we do not lose as much accuracy as might be expected when creating the SNN.

3. On both test problems, the shape of the SNN bound is somewhat similar to the shape of the true error rate. In particular, the local minima in the SNN bound occur roughly where the local minima in the true error rates occur. The SNN bound may weakly predict the overfitting points of the SNN and NN nets.

The comparison between the neural network bound and the stochastic neural network bound is not quite "fair" due to the form of the bound. In particular, the stochastic neural network bound can never return a value greater than "always err". This implies that when the bound is near the value of "1", it is difficult to judge how rapidly extra examples will improve the stochastic neural network bound. We can judge the sample complexity of the stochastic bound by plotting the value of the numerator in equation 1. Figure 3 plots the complexity versus the number of pattern presentations in training. In this figure, we

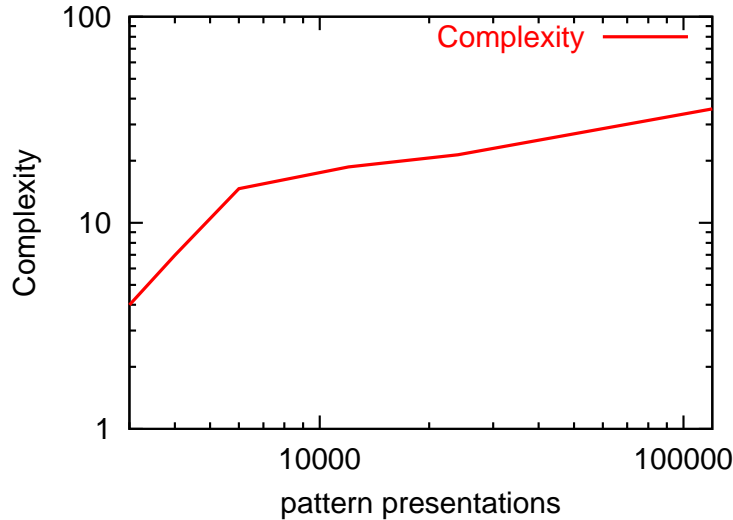

Figure 3: We plot the "complexity" of the stochastic network model (numerator of equation 1) vs. training epoch. Note that the complexity increases with more training as expected and stays below 100, implying nonvacuous bounds on a training set of size 100.

observe the expected result: the "complexity" (numerator of equation 1) increases with more training and is significantly less than the number of examples (100).

The stochastic bound is a radical improvement on the neural network bound but it is not yet a perfectly tight bound. Given that we do not have a perfectly tight bound, one important consideration arises: does the minimum of the stochastic bound predict the minimum of the true error rate (as predicted by a large holdout dataset). In particular, can we use the stochastic bound to determine when we should cease training? The stochastic bound depends upon (1) the complexity which increases with training time and (2) the training error which decreases with training time. This dependence results in a minima which occurs at approximately 12000 pattern presentations for both of our test problems. The point of minimal true error (for the stochastic and deterministic neural networks) occurs at approximately 6000 pattern presentations for the synthetic problem, and at about 18000 pattern presentations for the ADULT problem, indicating that the stochastic bound weakly predicts the point of minimum error. The neural network bound has no such minimum.

Is the choice of 1-5% increased empirical error optimal? In general, the "optimal" choice of the extra error rate depends upon the learning problem. Since the stochastic neural network bound (corollary 2.4) holds for all multidimensional gaussian distributions, we are free to optimize the choice of distribution in anyway we desire. Figure 4 shows the resulting bound for different choices of posterior $Q$. The bound has a minimum at $0.03$ extra error indicating that our initial choices of $0.01$ and $0.05$ are in the right ballpark, and $0.05$ may be unnecessarily large. Larger differences in empirical error rate such as $0.05$ are easier to obtain reliably with fewer samples from the stochastic neural net, but we have not had difficulty using as few as 100 samples from the SNN with as small as a 1% increase in empirical error. Also note that the complexity always decreases with increasing entropy in the distribution of our stochastic neural net. The existence of a minimum in Figure 4 is the "right" behaviour: the increased empirical error rate is significant in the calculation of the true error bound.

## 4   Conclusion

We have applied a PAC-Bayes bound for the true error rate of a stochastic neural network. The stochastic neural network bound results in a radically tighter $(2 - 3$ orders of mag-

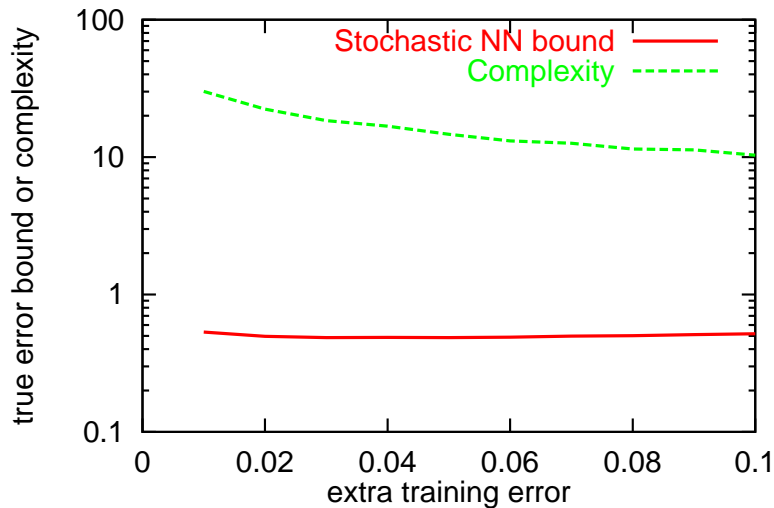

Figure 4: Plot of the stochastic neural net (SNN) bound for "posterior" distributions chosen according to the extra empirical error they introduce.

nitude) bound on the true error rate of a classifier while increasing the empirical and true error rates only a small amount.

Although, the stochastic neural net bound is not completely tight, it is not vacuous with just $100 - 200$ examples and the minima of the bound weakly predicts the point where overtraining occurs.

The results with two datasets (one synthetic and one from UCI) are extremely promising—the bounds are *orders of magnitude* better. Our next step will be to test the method on more datasets using a greater variety of net architectures to insure that the bounds remain tight. In addition, there remain many opportunities for improving the application of the bound. For example, it is possible that shifting the weights when finding a maximum acceptable variance will result in a tighter bound. Also, we have not taken into account symmetries within the network which would allow for a tighter bound calculation.

## References

[1] Peter Bartlett, "The Sample Complexity of Pattern Classification with Neural Networks: The Size of the Weights is More Important than the Size of the Network", IEEE Transactions on Information Theory, Vol. 44, No. 2, March 1998.

[2] V. Koltchinskii and D. Panchenko, "Empirical Margin Distributions and Bounding the Generalization Error of Combined Classifiers", preprint, http://citeseer.nj.nec.com/386416.html

[3] John Langford and Matthias Seeger, "Bounds for Averaging Classifiers." CMU tech report, 2001.

[4] David MacKay, "Probable Networks and Plausible Predictions - A Review of Practical Bayesian Methods for Supervised Neural Networks", ??

[5] David McAllester, "Some PAC-Bayes bounds", COLT 1999.
